# Synchronization, oscillations, and $1/f$ noise in networks of spiking neurons

**Martin Stemmler, Marius Usher, and Christof Koch**
Computation and Neural Systems, 139-74
California Institute of Technology
Pasadena, CA 91125

**Zeev Olami**
Dept. of Chemical Physics
Weizmann Institute of Science
Rehovot 76100, Israel

## Abstract

We investigate a model for neural activity that generates long range temporal correlations, $1/f$ noise, and oscillations in global activity. The model consists of a two-dimensional sheet of leaky integrate-and-fire neurons with feedback connectivity consisting of local excitation and surround inhibition. Each neuron is independently driven by homogeneous external noise. Spontaneous symmetry breaking occurs, resulting in the formation of "hotspots" of activity in the network. These localized patterns of excitation appear as clusters that coalesce, disintegrate, or fluctuate in size while simultaneously moving in a random walk constrained by the interaction with other clusters. The emergent cross-correlation functions have a dual structure, with a sharp peak around zero on top of a much broader hill. The power spectrum associated with single units shows a $1/f$ decay for small frequencies and is flat at higher frequencies, while the power spectrum of the spiking activity averaged over many cells—equivalent to the local field potential—shows no $1/f$ decay but a prominent peak around 40 Hz.

# 1    The model

The model consists of a 100-by-100 lattice of integrate-and-fire units with cyclic lattice boundary conditions. Each unit represents the nerve cell membrane as a simple RC circuit ($\tau = 20$ msec) with the addition of a reset mechanism; the refractory period $T_{ref}$ is equal to one iteration step (1 msec).

Units are connected to each other within the layer by local excitatory and inhibitory connections in a *center-surround* pattern. Each unit is excitatorily connected to $N = 50$ units chosen from a Gaussian probability distribution of $\sigma = 2.5$ (in terms of the lattice constant), centered at the unit's position $N$ inhibitory connections per unit are chosen from a uniform probability distribution on a ring eight to nine lattice constants away.

Once a unit reaches the threshold voltage, it emits a pulse that is transmitted in one iteration (1 msec) to connected neighboring units, and the potential is reset by subtracting the threshold from resting potential.

$$V_i(t + 1) = (\exp(-1/\tau)V_i(t) + I_i(t))\,\theta[V_{th} - V(t)]. \tag{1}$$

$I_i$ is the input current, which is the sum of lateral currents from presynaptic units and external current. The lateral current leads to an increase (decrease) in the membrane potential of excitatory (inhibitorily ) connected cells. The weight of the excitation and inhibition, in units of voltage threshold, is $\frac{\alpha}{N}$ and $\beta\frac{\alpha}{N}$. The values $\alpha = 1.275$ and $\beta = 0.67$ were used for simulations. The external input is modeled independently for each cell as a Poisson process of excitatory pulses of magnitude $1/N$, arriving at a mean rate $\lambda_{ext}$. Such a simple cellular model mimics reasonably well the discharge patterns of cortical neurons [Bernander et al., 1994, Softky and Koch, 1993].

# 2    Dynamics and Pattern Formation

In the mean-field approximation, the firing rate of an integrate-and-fire unit is a function of the input current [Amit and Tsodyks, 1991] given by

$$f(I) = (T_{ref} - \tau \ln[1 - 1/(I\,\tau)])^{-1}, \tag{2}$$

where $T_{ref}$ is the refractory period and $\tau$ the membrane time constant.

In this approximation, the dynamics associated with eq. 1 simplify to

$$\frac{dI_i}{dt} = -I_i + \sum_j W_{ij} f(I_j) + I_i^{ext}, \tag{3}$$

where $W_{ij}$ represents the connection strength matrix from unit j to unit i.

Homogeneous firing activity throughout the network will result as long as the connectivity pattern satisfies $\tilde{W}(k) - 1 < 0$ for all $k$, where $\tilde{W}(k)$ is the Fourier transform of $W_{ij}$. As one increases the total strength of lateral connectivity, clusters of high firing activity develop. These clusters form a hexagonal grid across the network; for even stronger lateral currents, the clusters merge to form stripes.

The transition from a homogeneous state to hexagonal clusters to stripes is generic to many nonequilibrium systems in fluid mechanics, nonlinear optics, reaction-diffusion systems, and biology. (The classic theory for fluid mechanics was first

developed by [Newell and Whitehead, 1969], see [Cross and Hohenberg, 1993] for an extensive review. Cowan (1982) was the first to suggest applying the techniques of fluid mechanics to neural systems.)

The richly varied dynamics of the model, however, can **not** be captured by a mean-field description. Clusters in the quasi-hexagonal state coalesce, disintegrate, or fluctuate in size while simultaneously moving in a random walk constrained by the interaction with other clusters.

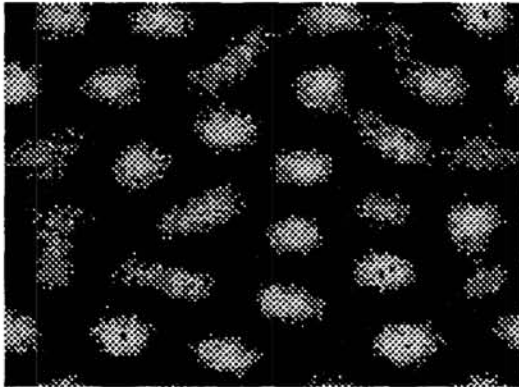
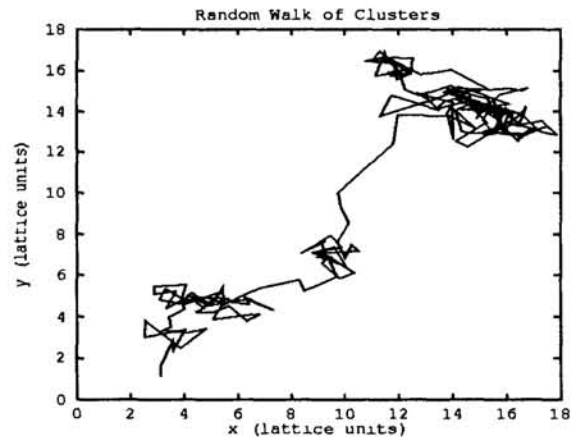

Figure 1: On the left, the summed firing activity for the network over 50 msec of simulation is shown. Lighter shades denote higher firing rates (maximum firing rate 120 Hz). Note the nearly hexagonal pattern of clusters or "hotspots" of activity. On the right, we illustrate the motion of a typical cluster. Each vertex in the graph represents a tracked cluster's position averaged over 50 msec. Repulsive interactions with surrounding clusters generally constrain the motion to remain within a certain radius. This vibratory motion of a cluster is occasionally punctuated by longer-range diffusion.

Statistical fluctuations, diffusion and synchronization of clusters, and noise in the external input driving the system lead to $1/f$-noise dynamics, long-range correlations, and oscillations in the local field potential. These issues shall be explored next.

## 3    $1/f$ Noise

The power spectra of spike trains from individual units are similar to those published in the literature for nonbursting cells in area MT in the behaving monkey [Bair et al., 1994]. Power spectra were generally flat for all frequencies above 100 Hz. The effective refractory period present in an integrate-and-fire model introduces a dip at low frequencies (also seen in real data). Most noteworthy is the $1/f^{0.8}$ component in the power spectrum at low frequencies. Notice that in order to see such a decay for very low frequencies in the spectrum, single units must be recorded for on the order of 10-100 sec, longer than the recording time for a typical trial in neurophysiology.

We traced a cluster of neuronal activity as it diffused through the system, and

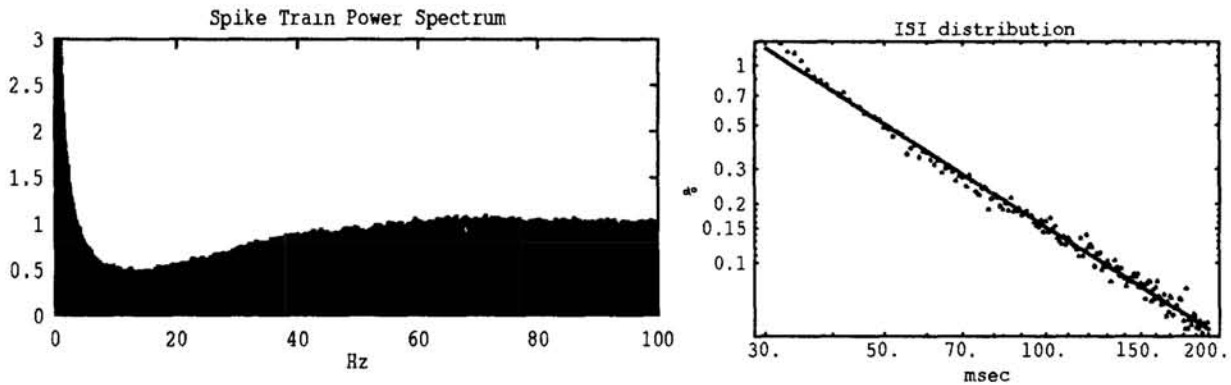

Figure 2: Typical power spectrum and ISI distribution of single units over 400 sec of simulation. At low frequencies, the power spectrum behaves as $f^{-0.8\pm0.017}$ up to a cut-off frequency of $\approx 8$ Hz. The ISI distribution on the right is shown on a log-log scale. The ISI histogram decays as a power law $P(t) \propto t^{-1.70\pm0.02}$ between 25 and 300 msec. In contrast, a system with randomized network connections will have a Poisson-distributed ISI histogram which decays exponentially.

measured the ISI distribution at a fixed point relative to the cluster center. In the cluster frame of reference, activity should remain fairly constant, so we expect and do find an interspike interval (ISI) distribution with a single characteristic relaxation time:

$$P_r(t) = \lambda(r) \exp(-t\lambda(r)),$$

where the firing rate $\lambda(r)$ is now only a function of the distance $r$ in cluster coordinates. Thus $P_r(t)$ is always *Poisson* for fixed $r$.

If a cluster diffuses slowly compared to the mean interspike interval, a unit at a fixed position samples many ISI distributions of varying $\lambda(r)$ as the cluster moves. The ISI distribution in the fixed frame reference is thus

$$P(t) = \int \lambda(r)^2 \exp(-t\,\lambda(r))dr. \tag{4}$$

Depending on the functional form of $\lambda(r)$, $P(t)$ (the ISI distribution for a unit at a *fixed* position) will decay as a power law, and *not* as an exponential. Empirically, the distribution of firing rates as a function of $r$ can be approximated (roughly) by a Gaussian. A Gaussian $\lambda(r)$ in eq. 4 leads to $P(t) \sim t^{-2}$ for $t$ at long times. In turn, a power-law (fractal) $P(t)$ generates $1/f$ noise (see Table 1).

## 4   Long-Range Cross-Correlations

Excitatory cross-correlation functions for units separated by small distances consist of a sharp peak at zero mean time delay followed by a slower decay characterized by a power law with exponent $-0.21$ until the function reaches an asymptotic level. Nelson *et al.* (1992) found this type of cross-correlation between neurons–a "castle on a hill"–to be the most common form of correlation in cat visual cortex. Inhibitory

cross-correlations show a slight dip that is much less pronounced than the sharp excitatory peak at short time-scales.

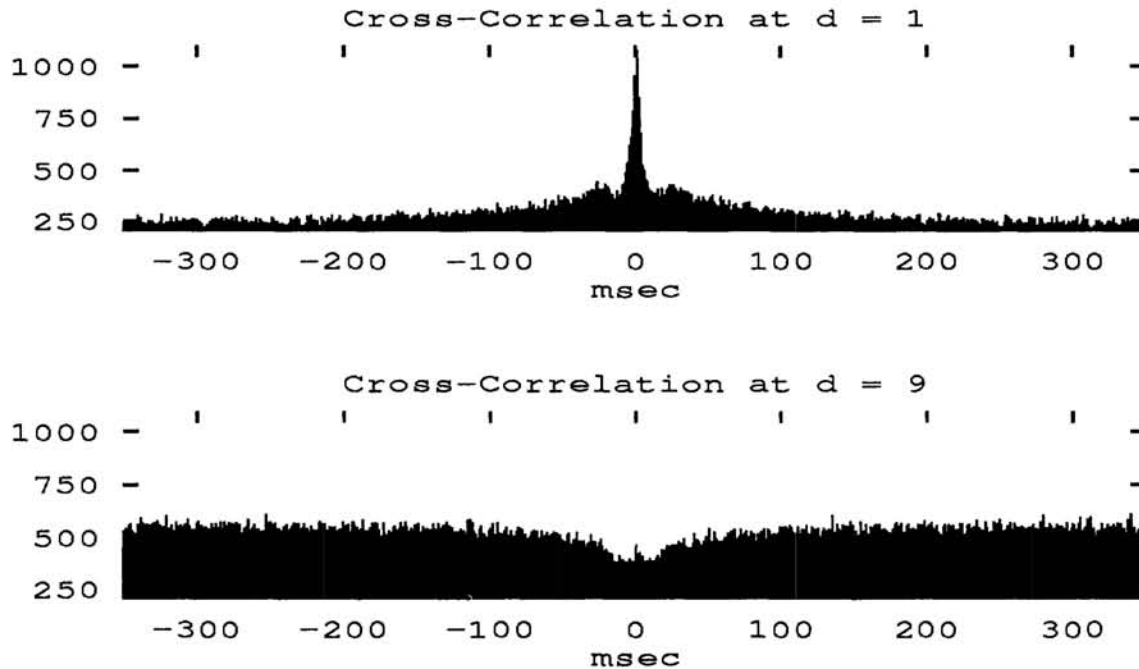

Figure 3: Cross-correlation functions between cells separated by $d$ units of the lattice. Given the *center-surround* geometry of connections, the upper curve corresponds to mutually excitatory coupling and the lower to mutually inhibitory coupling. Correlations decay as $1/t^{0.21}$, consistent with a power spectrum of single spike trains that behaves as $1/f^{0.8}$.

Since correlations decay slowly in time due to the small exponent of the power, long temporal fluctuations in the firing rate result, as the $1/f$-type power spectra of single spike trains demonstrate. These fluctuations in turn lead to high variability in the number of events over a fixed time period.

In fact, the decay in the auto-correlation and power spectrum, as well as the rise in the variability in the number of events, can be related back to the slow decay in the interspike interval (ISI) distribution. If the ISI distribution decays as a power law $P(t) \sim t^{-\nu}$, then the point process giving rise to it is fractal with a dimension $D = \nu - 1$ [Mandelbrot, 1983]. Assuming that the simulation model can be described as a fully ergodic renewal process, all these quantities will scale together [Cox and Lewis, 1966, Teich, 1989, Lowen and Teich, 1993, Usher et al., 1994]:

Table 1: Scaling Relations and Empirical Results

| Var($N$) | Auto-correlation | Power Spectrum | ISI Distribution |
|---|---|---|---|
| Var(N) $\sim N^\nu$ | $A(t) \sim t^{\nu-2}$ | $S(f) \sim f^{-\nu+1}$ | $P(t) \sim t^{-\nu}$ |
| Var(N) $\sim N^{1.54}$ | $A(t) \sim t^{-0.21}$ | $S(f) \sim f^{-0.81}$ | $P(t) \sim t^{-1.70}$ |

These relations will be only approximate if the process is nonrenewal or nonergodic, or if power-laws hold over a limited range. The process in the model is clearly non-renewal, since the presence of a cluster makes consecutive short interspike intervals for units within that cluster more likely than in a renewal process. Hence, we expect some (slight) deviations from the scaling relations outlined above.

## 5     Cluster Oscillations and the Local Field Potential

The interplay between the recurrent excitation that leads to nucleation of clusters and the "firewall" of inhibition that restrains activity causes clusters of high activity to oscillate in size. Fig 4 is the power spectrum of ensemble activity over the size of a typical cluster.

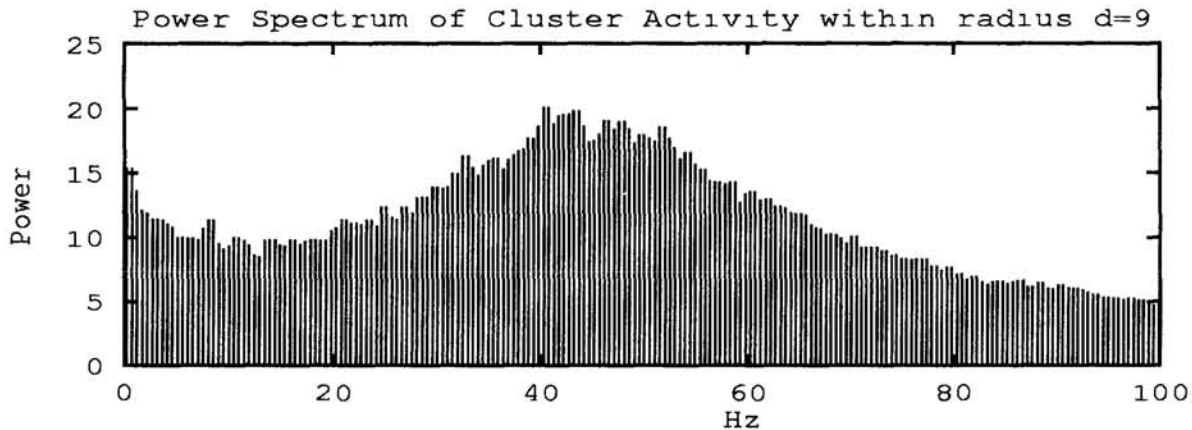

Figure 4: Power spectrum of the summed spiking activity over a circular area the size of a single cluster (with a radius of 9 lattice constants) recorded from a fixed point on the lattice for 400 seconds. Note the prominent peak centered at 43 Hz and the loss of the $1/f$ component seen in the single unit power spectra (Fig. 2).

These oscillations can be understood by examining the cross-correlations between cells. By the Wiener-Khinchin theorem, the power spectrum of cluster activity is the Fourier transform of the signal's auto-correlation. Since the cluster activity is the sum of all single-unit spiking activity within a cluster of $N$ cells, the autocorrelation of the cluster spiking activity will be the sum of $N$ auto-correlations functions of the

individual cells and $N \times (N - 1)$ cross-correlation functions among individual cells within the cluster. The ensemble activity is thus dominated by cross-correlations.

In general, the excitatory "castles" are sharp relative to the broad dip in the cross-correlation due to inhibition (see Fig. 3). In Fourier space, these relationships are reversed: broader Fourier transforms of excitatory cross-correlations are paired with narrower Fourier transforms of inhibitory cross-correlations. Superposition of such transforms leads to a peak in the 30-70 Hz range and cancellation of the $1/f$ component which was present the single unit power spectrum.

Interestingly, the power spectra of spike trains of individual cells within the network (Fig. 2) show no evidence of a peak in this frequency band. Diffusion of clusters disrupts any phase relationship between single unit firing and ensemble activity.

The ensemble activity corresponds to the local field potential in neurophysiological recordings. While oscillations between 30 and 90 Hz have often been seen in the local field potential (or sometimes even in the EEG) measured in cortical areas in the anesthetized or awake cat and monkey, these oscillations are frequently not or only weakly visible in multi- or single-unit data (e.g., [Eeckman and Freeman, 1990, Kreiter and Singer, 1992, Gray et al., 1990, Eckhorn et al., 1993]). We here offer a general explanation for this phenomenon.

**Acknowledgments:** We are indebted to William Softky, Wyeth Bair, Terry Sejnowski, Michael Cross, John Hopfield, and Ernst Niebur, for insightful discussions. Our research was supported by a Myron A. Bantrell Research Fellowship, the Howard Hughes Medical Institute, the National Science Foundation, the Office of Naval Research and the Air Force Office of Scientific Research.

# References

[Amit and Tsodyks, 1991] Amit, D. J. and Tsodyks, M. V. (1991). Quantitative study of attractor neural network retrieving at low rates:1. substrate spikes, rates and neuronal gain. *Network Com.*, 2(3):259–273.

[Bair et al., 1994] Bair, W., Koch, C., Newsome, W., and Britten, K. (1994). Power spectrum analysis of MT neurons in the behaving monkey. *J. Neurosci.*, in press.

[Bernander et al., 1994] Bernander, O., Koch, C., and Usher, M. (1994). The effect of synchronized inputs at the single neuron level. *Neural Computation*, in press.

[Cowan, 1982] Cowan, J. D. (1982). Spontaneous symmetry breaking in large scale nervous activity. *Int. J. Quantum Chemistry*, 22:1059–1082.

[Cox and Lewis, 1966] Cox, D. and Lewis, P. A. W. (1966). *The Statistical Analysis of Series of Events*. Chapman and Hall, London.

[Cross and Hohenberg, 1993] Cross, M. C. and Hohenberg, P. C. (1993). Pattern formation outside of equilibrium. *Rev. Mod. Phys.*, 65(3):851–1112.

[Eckhorn et al., 1993] Eckhorn, R., Frien, A., Bauer, R., Woelbern, T., and Harald, K. (1993). High frequency (60-90 hz) oscillations in primary visual cortex of awake monkey. *Neuroreport*, 4:243–246.

[Eeckman and Freeman, 1990] Eeckman, F. and Freeman, W. (1990). Correlations between unit firing and EEG in the rat olfactory system. *Brain Res.*, 528(2):238–244.

[Gray et al., 1990] Gray, C. M., Engel, A. K., König, P., and Singer, W. (1990). Stimulus dependent neuronal oscillations in cat visual cortex: receptive field properties and feature dependence. *Europ. J. Neurosci.*, 2:607–619.

[Kreiter and Singer, 1992] Kreiter, A. K. and Singer, W. (1992). Oscillatory neuronal responses in the visual cortex of the awake macaque monkey. *Europ. J. Neurosci.*, 4:369–375.

[Lowen and Teich, 1993] Lowen, S. B. and Teich, M. C. (1993). Fractal renewal processes generate 1/f noise. *Phys. Rev. E*, 47(2):992–1001.

[Mandelbrot, 1983] Mandelbrot, B. B. (1983). *The fractal geometry of nature*. W. H. Freeman, New York.

[Nelson et al., 1992] Nelson, J. I., Salin, P. A., Munk, M. H.-J., Arzi, M., and Bullier, J. (1992). Spatial and temporal coherence in cortico-cortical connections: A cross-correlation study in areas 17 and 18 in the cat. *Visual Neuroscience*, 9:21–38.

[Newell and Whitehead, 1969] Newell, A. C. and Whitehead, J. A. (1969). Finite bandwidth, finite amplitude convection. *J. Fluid Mech.*, 38:279–303.

[Softky and Koch, 1993] Softky, W. R. and Koch, C. (1993). The highly irregular firing of cortical cells is inconsistent with temporal integration of random EPSPs. *J. Neurosci.*, 13(1):334–350.

[Teich, 1989] Teich, M. C. (1989). Fractal character of the auditory neural spike train. *IEEE Trans. Biomed. Eng.*, 36(1):150–160.

[Usher et al., 1994] Usher, M., Stemmler, M., Koch, C., and Olami, Z. (1994). Network amplification of local fluctuations causes high spike rate variability, fractal firing patterns, and oscillatory local field potentials. *Neural Computation*, in press.


